# When is an Integrate-and-fire Neuron like a Poisson Neuron?

**Charles F. Stevens**
Salk Institute MNL/S
La Jolla, CA 92037
cfs@salk.edu

Anthony Zador
Salk Institute MNL/S
La Jolla, CA 92037
zador@salk.edu

## Abstract

In the Poisson neuron model, the output is a rate-modulated Poisson process (Snyder and Miller, 1991); the time varying rate parameter $r(t)$ is an instantaneous function $G[.]$ of the stimulus, $r(t) = G[s(t)]$. In a Poisson neuron, then, $r(t)$ gives the instantaneous firing rate—the instantaneous probability of firing at any instant $t$—and the output is a stochastic function of the input. In part because of its great simplicity, this model is widely used (usually with the addition of a refractory period), especially in *in vivo* single unit electrophysiological studies, where $s(t)$ is usually taken to be the value of some sensory stimulus. In the integrate-and-fire neuron model, by contrast, the output is a filtered and thresholded function of the input: the input is passed through a low-pass filter (determined by the membrane time constant $\tau$) and integrated until the membrane potential $v(t)$ reaches threshold $\theta$, at which point $v(t)$ is reset to its initial value. By contrast with the Poisson model, in the integrate-and-fire model the ouput is a deterministic function of the input. Although the integrate-and-fire model is a caricature of real neural dynamics, it captures many of the qualitative features, and is often used as a starting point for conceptualizing the biophysical behavior of single neurons. Here we show how a slightly modified Poisson model can be derived from the integrate-and-fire model with noisy inputs $y(t) = s(t) + n(t)$. In the modified model, the transfer function $G[.]$ is a sigmoid (erf) whose shape is determined by the noise variance $\sigma_n^2$. Understanding the equivalence between the dominant *in vivo* and *in vitro* simple neuron models may help forge links between the two levels.

# 1 Introduction

In the Poisson neuron model, the output is a rate-modulated Poisson process; the time varying rate parameter $r(t)$ is an instantaneous function $G[.]$ of the stimulus, $r(t) = G[s(t)]$. In a Poisson neuron, then, $r(t)$ gives the instantaneous firing rate—the instantaneous probability of firing at any instant $t$—and the output is a stochastic function of the input. In part because of its great simplicity, this model is widely used (usually with the addition of a refractory period), especially in *in vivo* single unit electrophysiological studies, where $s(t)$ is usually taken to be the value of some sensory stimulus.

In the integrate-and-fire neuron model, by contrast, the output is a filtered and thresholded function of the input: the input is passed through a low-pass filter (determined by the membrane time constant $\tau$) and integrated until the membrane potential $v(t)$ reaches threshold $\theta$, at which point $v(t)$ is reset to its initial value. By contrast with the Poisson model, in the integrate-and-fire model the ouput is a deterministic function of the input. Although the integrate-and-fire model is a caricature of real neural dynamics, it captures many of the qualitative features, and is often used as a starting point for conceptualizing the biophysical behavior of single neurons (Softky and Koch, 1993; Amit and Tsodyks, 1991; Shadlen and Newsome, 1995; Shadlen and Newsome, 1994; Softky, 1995; DeWeese, 1995; DeWeese, 1996; Zador and Pearlmutter, 1996).

Here we show how a slightly modified Poisson model can be derived from the integrate-and-fire model with noisy inputs $y(t) = s(t) + n(t)$. In the modified model, the transfer function $G[.]$ is a sigmoid (erf) whose shape is determined by the noise variance $\sigma_n^2$. Understanding the equivalence between the dominant *in vivo* and *in vitro* simple neuron models may help forge links between the two levels.

# 2 The integrate-and-fire model

Here we describe the the forgetful leaky integrate-and-fire model. Suppose we add a signal $s(t)$ to some noise $n(t)$,

$$y(t) = n(t) + s(t),$$

and threshold the sum to produce a spike train

$$z(t) = \mathcal{F}[s(t) + n(t)],$$

where $\mathcal{F}$ is the thresholding functional and $z(t)$ is a list of firing times generated by the input. Specifically, suppose the voltage $v(t)$ of the neuron obeys

$$\dot{v}(t) = -\frac{v(t)}{\tau} + y(t) \tag{1}$$

where $\tau$ is the membrane time constant. We assume that the noise $n(t)$ has 0-mean and is white with variance $\sigma_n^2$. Thus $y(t)$ can be thought of as a Gaussian white process with variance $\sigma_n^2$ and a time-varying mean $s(t)$. If the voltage reaches the threshold $\theta_0$ at some time $t$, the neuron emits a spike at that time and resets to the initial condition $v_0$. This is therefore a 5 parameter model: the membrane time constant $\tau$, the mean input signal $\mu$, the variance of the input signal $\sigma^2$, the threshold $\theta$, and the reset value $v_0$. Of course, if $n(t) = 0$, we recover a purely deterministic integrate-and-fire model.

In order to forge the link between the integrate-and-fire neuron dynamics and the Poisson model, we will treat the firing times $T$ probabilistically. That is, we will express the output of the neuron to some particular input $s(t)$ as a conditional distribution $p(T|s(t))$, *i.e.* the probability of obtaining any firing time $T$ given some particular input $s(t)$.

Under these assumptions, $p(T)$ is given by the first passage time distribution (FPTD) of the Ornstein-Uhlenbeck process (Uhlenbeck and Ornstein, 1930; Tuckwell, 1988). This means that the time evolution of the voltage prior to reaching threshold is given by the Fokker-Planck equation (FPE),

$$\frac{\partial}{\partial t}g(t,v) = \frac{\sigma_y^2}{2}\frac{\partial^2}{\partial v^2}g(t,v) - \frac{\partial}{\partial v}[(s(t) - \frac{v(t)}{\tau})g(t,v)], \tag{2}$$

where $\sigma_y = \sigma_n$ and $g(t,v)$ is the distribution at time $t$ of voltage $-\infty < v \leq \theta_0$. Then the first passage time distribution is related to $g(v,t)$ by

$$p(T) = -\frac{\partial}{\partial t}\int_{-\infty}^{\theta_0} g(t,v)dv. \tag{3}$$

The integrand is the fraction of all paths that have not yet crossed threshold. $p(T)$ is therefore just the interspike interval (ISI) distribution for a given signal $s(t)$. A general eigenfunction expansion solution for the ISI distribution is known, but it converges slowly and its terms offer little insight into the behavior (at least to us).

We now derive an expression for the probability of crossing threshold in some very short interval $\Delta t$, starting at some $v$. We begin with the "free" distribution of $g$ (Tuckwell, 1988): the probability of the voltage jumping to $v'$ at time $t' = t + \Delta t$, given that it was at $v$ at time $t$, assuming von Neumann boundary conditions at plus and minus infinity,

$$g(t',v'|t,v) = \frac{1}{\sqrt{2\pi\, q(\Delta t;\sigma_y)}}\exp\left[-\frac{(v' - m(\Delta t;\sigma_y))^2}{2\, q(\Delta t;\sigma_y)}\right], \tag{4}$$

with

$$q_\Delta = \sigma_y^2\tau(1 - e^{-2\,\Delta t/\tau})$$

and

$$m(\Delta t) = ve^{-\Delta t/\tau} + s(t) * \tau(1 - e^{-\Delta t/\tau}),$$

where $*$ denotes convolution. The free distribution is a Gaussian with a time-dependent mean $m(\Delta t)$ and variance $q(\Delta t;\sigma_y)$. This expression is valid for all $\Delta t$. The probability of making a jump

$$\Delta v = v' - v$$

in a short interval $\Delta t \ll \tau$ depends only on $\Delta v$ and $\Delta t$,

$$g_\Delta(\Delta t, \Delta v; \sigma_y) = \frac{1}{\sqrt{2\pi\, q_\Delta(\sigma_y)}}\exp\left[-\frac{\Delta v^2}{2\, q_\Delta(\sigma_y)}\right]. \tag{5}$$

For small $\Delta t$, we expand to get

$$q_\Delta(\sigma_y) \approx 2\sigma_y^2\Delta t,$$

which is independent of $\tau$, showing that the leak can be neglected for short times.

Now the probability $P_\Delta$ that the voltage exceeds threshold in some short $\Delta t$, given that it started at $v$, depends on how far $v$ is from threshold; it is

$$\Pr[v + \Delta v \geq \theta] = \Pr[\Delta v \geq \theta - v].$$

Thus

$$
\begin{aligned}
P_\Delta &= \int_{\theta-v}^{\infty} dv g_\Delta(\Delta t, v; \sigma_y) \\
&= \frac{1}{2}\text{erfc}\left(\frac{\theta - v}{\sqrt{2q_\Delta(\sigma_y)}}\right) \\
&\approx \frac{1}{2}\text{erfc}\left(\frac{\theta - v}{\sigma_y\sqrt{2\Delta t}}\right)
\end{aligned}
\tag{6}
$$

where $\text{erfc}(x) = 1 - \frac{2}{\sqrt{\pi}}\int_0^x e^{-t^2}dt$ goes from $[2:0]$. This then is the key result: it gives the instantaneous probability of firing as a function of the instantaneous voltage $v$. erfc is sigmoidal with a slope determined by $\sigma_y$, so a smaller noise yields a steeper (more deterministic) transfer function; in the limit of 0 noise, the transfer function is a step and we recover a completely deterministic neuron.

Note that $P_\Delta$ is actually an instantaneous function of $v(t)$, not the stimulus itself $s(t)$. If the noise is large compared with $s(t)$ we must consider the distribution $g_s(v, t; \sigma_y)$ of voltages reached in response to the input $s(t)$:

$$
\begin{aligned}
P_Y(t) &= \int_{-\infty}^{\theta} g_s(\gamma, t; \sigma_y) \int_{\theta-\gamma}^{\infty} g_\Delta(\Delta t, \eta; \sigma_y)d\eta d\gamma \\
&= \int_{-\infty}^{\theta} g_s(\gamma, t; \sigma_y) \left[\frac{1}{2}\text{erfc}\left(\frac{\theta - \gamma}{\sqrt{2q_\Delta(\sigma_y)}}\right)\right] d\gamma
\end{aligned}
\tag{7}
$$

## 3  Ensemble of Signals

What if the inputs $s(t)$ are themselves drawn from an ensemble? If their distribution is also Gaussian and white with mean $\mu$ and variance $\sigma_s^2$, and if the firing rate is low ($E[T] \gg \tau$), then the output spike train is Poisson. Why is firing Poisson only in the slow firing limit? The reason is that, by assumption, immediately following a spike the membrane potential resets to 0; it must then rise (assuming $\mu > 0$) to some asymptotic level that is independent of the initial conditions. During this rise the firing rate is lower than the asymptotic rate, because on average the membrane is farther from threshold, and its variance is lower. The rate at which the asymptote is achieved depends on $\tau$. In the limit as $t \gg \tau$, some asymptotic distribution of voltage $q_\infty(v)$, is attained. Note that if we make the reset $v_0$ stochastic, with a distribution given by $q_\infty(v)$, then the firing probability would be the same even immediately after spiking, and firing would be Poisson for all firing rates.

A Poisson process is characterized by its mean alone. We therefore solve the FPE (eq. 2) for the steady-state by setting $\frac{\partial}{\partial t}g(t, v) = 0$ (we consider only threshold crossings from initial values $t \gg \tau$; neglecting the early events results in only a small error, since we have assumed $E\{T\} \gg \tau$). Thus with the absorbing boundary

at $\theta$ the distribution at time $t \gg \tau$ (given here for $\mu = 0$) is

$$g_\infty(v; \sigma_y) = k_1 \left(1 - k_2 \mathrm{erfi}\left[\frac{v}{\sigma_y \sqrt{\tau}}\right]\right) \exp\left[\frac{-v^2}{\sigma_y^2 \tau}\right], \tag{8}$$

where $\sigma_y^2 = \sigma_s^2 + \sigma_n^2$, $\mathrm{erfi}(z) = -i\mathrm{erf}(iz)$, $k_1$ determines the normalization (the sign of $k_1$ determines whether the solution extends to positive or negative infinity) and $k_2 = 1/\mathrm{erfi}(\theta/(\sigma_y \sqrt{\tau}))$ is determined by the boundary. The instantaneous Poisson rate parameter is then obtained through eq. (7),

$$
\begin{aligned}
P_Y &= \int_{-\infty}^{\theta} g_\infty(\gamma; \sigma_y) \int_{\theta-\gamma}^{\infty} g_\Delta(\Delta t, \eta; \sigma_y) d\eta \, d\gamma \\
&= \int_{-\infty}^{\theta} g_\infty(\gamma; \sigma_y) \left[\frac{1}{2}\mathrm{erfc}\left(\frac{\theta - \gamma}{\sqrt{2q_\Delta(\sigma_y)}}\right)\right] d\gamma
\end{aligned} \tag{9}
$$

Fig. 1 tests the validity of the exponential approximation. The top graph shows the ISI distribution near the "balance point", when the excitation is in balance with the inhibition and the membrane potential hovers just subthreshold. The bottom curves show the ISI distribution far below the balance point. In both cases, the exponential distribution provides a good approximation for $t \gg \tau$.

## 4  Discussion

The main point of this paper is to make explicit the relation between the Poisson and integrate-and-fire models of neuronal acitivity. The key difference between them is that the former is stochastic while the latter is deterministic. That is, given exactly the same stimulus, the Poisson neuron produces different spike trains on different trials, while the integrate-and-fire neuron produces exactly the same spike train each time. It is therefore clear that if some degree of stochasticity is to be obtained in the integrate-and-fire model, it must arise from noise in the stimulus itself.

The relation we have derived here is purely formal; we have intentionally remained agnostic about the deep issues of what is signal and what is noise in the inputs to a neuron. We observe nevertheless that although we derive a limit (eq. 9) where the spike train of an integrate-and-fire neuron is a Poisson process—*i.e.* the probability of obtaining a spike in any interval is independent of obtaining a spike in any other interval (except for very short intervals)—from the point of view of information processing it is a very different process from the purely stochastic rate-modulated Poisson neuron. In fact, in this limit the spike train is *deterministically Poisson* if $\sigma_y = \sigma_s$, *i.e.* when $n(t) = 0$; in this case the output is a purely deterministic function of the input, but the ISI distribution is exponential.

# References

Amit, D. and Tsodyks, M. (1991). Quantitative study of attractor neural network retrieving at low spike rates. i. substrate-spikes, rates and neuronal gain. *Network: Computation in Neural Systems*, 2:259–273.

DeWeese, M. (1995). *Optimization principles for the neural code*. PhD thesis, Dept of Physics, Princeton University.

DeWeese, M. (1996). Optimization principles for the neural code. In Hasselmo, M., editor, *Advances in Neural Information Processing Systems, vol. 8*. MIT Press, Cambridge, MA.

Shadlen, M. and Newsome, W. (1994). Noise, neural codes and cortical organization. *Current Opinion in Neurobiology*, 4:569–579.

Shadlen, M. and Newsome, W. (1995). Is there a signal in the noise? [comment]. *Current Opinion in Neurobiology*, 5:248–250.

Snyder, D. and Miller, M. (1991). *Random Point Processes in Time and Space*, $2^{nd}$ *edition*. Springer-Verlag.

Softky, W. (1995). Simple codes versus efficient codes. *Current Opinion in Neurobiology*, 5:239–247.

Softky, W. and Koch, C. (1993). The highly irregular firing of cortical cells is inconsistent with temporal integration of random epsps. *J. Neuroscience.*, 13:334–350.

Tuckwell, H. (1988). *Introduction to theoretical neurobiology (2 vols.)*. Cambridge.

Uhlenbeck, G. and Ornstein, L. (1930). On the theory of brownian motion. *Phys. Rev.*, 36:823–841.

Zador, A. M. and Pearlmutter, B. A. (1996). VC dimension of an integrate and fire neuron model. *Neural Computation*, 8(3). In press.

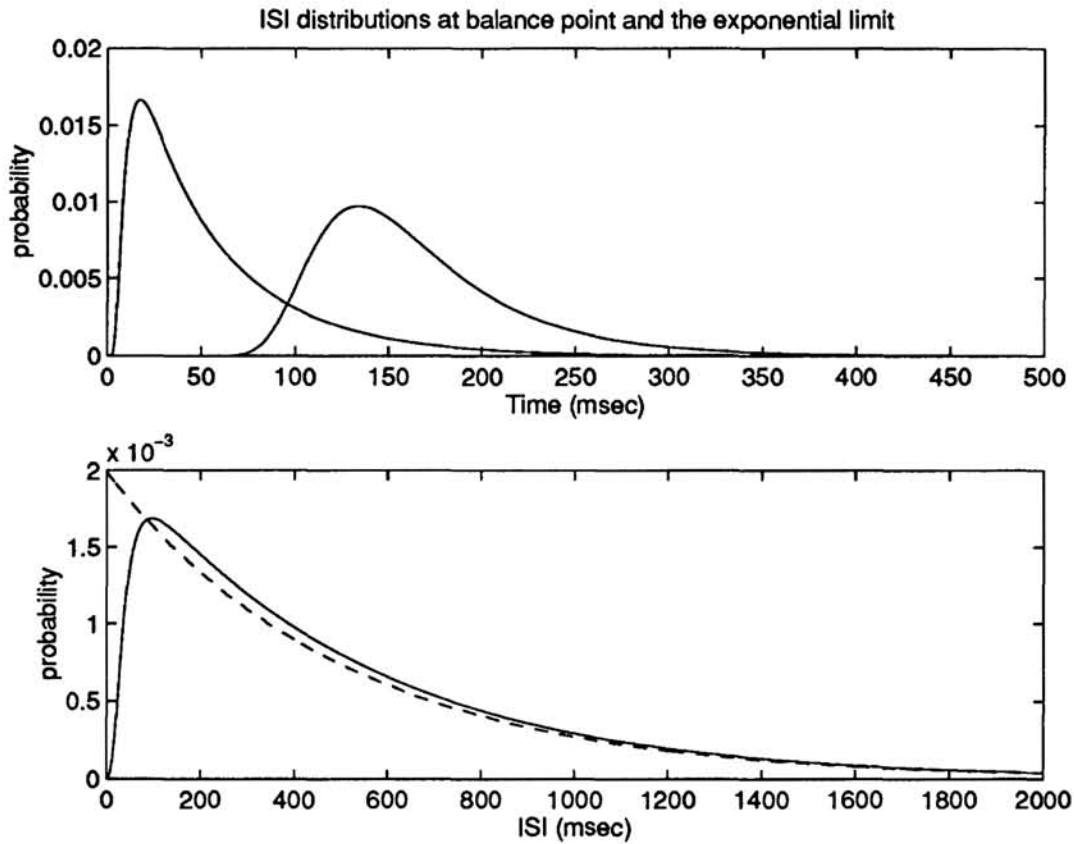

Figure 1: ISI distributions. (A; *top*) ISI distribution for leaky integrate-and-fire model at the balance point, where the asymptotic membrane potential is just subthreshold, for two values of the signal variance $\sigma^2$. Increasing $\sigma^2$ shifts the distribution to the left. For the left curve, the parameters were chosen so that $E\{T\} \approx \tau$, giving a nearly exponential distribution; for the *right* curve, the distribution would be hard to distinguish experimentally from an exponential distribution with a refractory period. ($\tau = 50$ msec; *left*: $E\{T\} = 166$ msec; *right*: $E\{T\} = 57$ msec). (B; *bottom*) In the subthreshold regime, the ISI distribution (*solid*) is nearly exponential (*dashed*) for intervals greater than the membrane time constant. ($\tau = 50$ msec; $E\{T\} = 500$ msec)